# More data means less inference: A pseudo-max approach to structured learning

**David Sontag**
Microsoft Research

**Ofer Meshi**
Hebrew University

**Tommi Jaakkola**
CSAIL, MIT

**Amir Globerson**
Hebrew University

## Abstract

The problem of learning to predict structured labels is of key importance in many applications. However, for general graph structure both learning and inference are intractable. Here we show that it is possible to circumvent this difficulty when the distribution of training examples is rich enough, via a method similar in spirit to pseudo-likelihood. We show that our new method achieves consistency, and illustrate empirically that it indeed approaches the performance of exact methods when sufficiently large training sets are used.

Many prediction problems in machine learning applications are *structured* prediction tasks. For example, in protein folding we are given a protein sequence and the goal is to predict the protein's native structure [14]. In parsing for natural language processing, we are given a sentence and the goal is to predict the most likely parse tree [2]. In these and many other applications, we can formalize the structured prediction problem as taking an input $x$ (e.g., primary sequence, sentence) and predicting $y$ (e.g., structure, parse) according to $y = \arg\max_{\hat{y} \in \mathcal{Y}} \theta \cdot \phi(x, \hat{y})$, where $\phi(x, y)$ is a function that maps any input and a candidate assignment to a feature vector, $\mathcal{Y}$ denotes the space of all possible assignments to the vector $y$, and $\theta$ is a weight vector to be learned.

This paper addresses the problem of *learning* structured prediction models from data. In particular, given a set of labeled examples $\{(x^m, y^m)\}_{m=1}^M$, our goal is to find a vector $\theta$ such that for each example $m$, $y^m = \arg\max_{y \in \mathcal{Y}} \theta \cdot \phi(x^m, y)$, i.e. one which *separates* the training data. For many structured prediction models, maximization over $\mathcal{Y}$ is computationally intractable. This makes it difficult to apply previous algorithms for learning structured prediction models, such as structured perceptron [2], stochastic subgradient [10], and cutting-plane algorithms [5], which require making a prediction at every iteration (equivalent to repeatedly solving an integer linear program).

Given training data, we can consider the space of parameters $\Theta$ that separate the data. This space can be defined by the intersection of a large number of linear inequalities. A recent approach to getting around the hardness of prediction is to use linear programming (LP) relaxations to approximate the maximization over $\mathcal{Y}$ [4, 6, 9]. However, separation with respect to a relaxation places stronger constraints on the parameters. The target solution, an integral vertex in the LP, must now distinguish itself also from possible fractional vertexes that arise due to the relaxation. The relaxations can therefore be understood as optimizing over an *inner bound* of $\Theta$. This set may be empty even if the training data is separable with exact inference [6]. Another obstacle to using LP relaxations for learning is that solving the LPs can be very slow.

In this paper we ask whether it is possible to learn while avoiding inference altogether. We propose a new learning algorithm, inspired by pseudo-likelihood [1], that optimizes over an *outer bound* of $\Theta$. Learning involves optimizing over only a small number of constraints per data point, and thus can be performed quickly, even for complex structured prediction models. We show that, if the data available for learning is "nice", this algorithm is *consistent*, i.e. it will find some $\theta \in \Theta$. This is an example of how having the right data can circumvent the hardness of learning for structured prediction.

We also investigate the limitations of the proposed method. We show that the problem of even deciding whether a given data set is separable is NP-hard, and thus learning in a strict sense is no easier than prediction. Thus, we should not expect for our algorithm, or any other polynomial time algorithm, to always succeed at learning from an arbitrary finite data set. To our knowledge, this is the first result characterizing the hardness of exact learning for structured prediction.

Finally, we show empirically that our algorithm allows us to successfully learn the parameters for both multi-label prediction and protein side-chain placement. The performance of the algorithm is improved as more data becomes available, as our theoretical results anticipate.

## 1 Pseudo-Max method

We consider the general structured prediction problem. The input space is denoted by $\mathcal{X}$ and the set of all possible assignments by $\mathcal{Y}$. Each $\boldsymbol{y} \in \mathcal{Y}$ corresponds to $n$ variables $y_1, \ldots, y_n$, each with $k$ possible states. The classifier uses a (given) function $\boldsymbol{\phi}(\boldsymbol{x}, \boldsymbol{y}) : \mathcal{X}, \mathcal{Y} \to \mathbb{R}^d$ and (learned) weights $\boldsymbol{\theta} \in \mathbb{R}^d$, and is defined as $\boldsymbol{y}(\boldsymbol{x}; \boldsymbol{\theta}) = \arg \max_{\hat{\boldsymbol{y}} \in \mathcal{Y}} f(\hat{\boldsymbol{y}}; \boldsymbol{x}, \boldsymbol{\theta})$ where $f$ is the discriminant function $f(\boldsymbol{y}; \boldsymbol{x}, \boldsymbol{\theta}) = \boldsymbol{\theta} \cdot \boldsymbol{\phi}(\boldsymbol{x}, \boldsymbol{y})$. Our analysis will focus on functions $\phi$ whose scope is limited to small sets of the $y_i$ variables, but for now we keep the discussion general.

Given a set of labeled examples $\{(\boldsymbol{x}^m, \boldsymbol{y}^m)\}_{m=1}^M$, the goal of the typical learning problem is to find weights $\boldsymbol{\theta}$ that correctly classify the training examples. Consider first the separable case. Define the set of separating weight vectors, $\Theta = \{\boldsymbol{\theta} \mid \forall m, \boldsymbol{y} \in \mathcal{Y}, \ f(\boldsymbol{y}^m; \boldsymbol{x}^m, \boldsymbol{\theta}) \geq f(\boldsymbol{y}; \boldsymbol{x}^m, \boldsymbol{\theta}) + e(\boldsymbol{y}, \boldsymbol{y}^m)\}$. $e$ is a loss function (e.g., zero-one or Hamming) such that $e(\boldsymbol{y}^m, \boldsymbol{y}^m) = 0$ and $e(\boldsymbol{y}, \boldsymbol{y}^m) > 0$ for $\boldsymbol{y} \neq \boldsymbol{y}^m$, which serves to rule out the trivial solution $\boldsymbol{\theta} = 0$.[1] The space $\Theta$ is defined by exponentially many constraints per example, one for each competing assignment.

In this work we consider a much simpler set of constraints where, for each example, we only consider the competing assignments obtained by modifying a single label $y_i$, while fixing the other labels to their value at $\boldsymbol{y}^m$. The *pseudo-max* set, which is an outer bound on $\Theta$, is given by

$$\Theta_{ps} = \{\boldsymbol{\theta} \mid \forall m, i, y_i, \ f(\boldsymbol{y}^m; \boldsymbol{x}^m, \boldsymbol{\theta}) \geq f(\boldsymbol{y}_{-i}^m, y_i; \boldsymbol{x}^m, \boldsymbol{\theta}) + e(y_i, y_i^m)\}. \tag{1}$$

Here $\boldsymbol{y}_{-i}^m$ denotes the label $\boldsymbol{y}^m$ without the assignment to $y_i$.

When the data is not separable, $\Theta$ will be the empty set. Instead, we may choose to minimize the hinge loss, $\ell(\boldsymbol{\theta}) = \sum_m \max_{\boldsymbol{y}} \left[ f(\boldsymbol{y}; \boldsymbol{x}^m, \boldsymbol{\theta}) - f(\boldsymbol{y}^m; \boldsymbol{x}^m, \boldsymbol{\theta}) + e(\boldsymbol{y}, \boldsymbol{y}^m) \right]$, which can be shown to be an upper bound on the training error [13]. When the data is separable, $\min_{\boldsymbol{\theta}} \ell(\boldsymbol{\theta}) = 0$. Note that regularization may be added to this objective.

The corresponding *pseudo-max* objective replaces the maximization over all of $\boldsymbol{y}$ with maximization over a single variable $y_i$ while fixing the other labels to their value at $\boldsymbol{y}^m$:[2,3]

$$\ell_{ps}(\boldsymbol{\theta}) = \sum_{m=1}^{M} \sum_{i=1}^{n} \max_{y_i} \left[ f(\boldsymbol{y}_{-i}^m, y_i; \boldsymbol{x}^m, \boldsymbol{\theta}) - f(\boldsymbol{y}^m; \boldsymbol{x}^m, \boldsymbol{\theta}) + e(y_i, y_i^m) \right]. \tag{2}$$

Analogous to before, we have $\min_{\boldsymbol{\theta}} \ell_{ps}(\boldsymbol{\theta}) = 0$ if and only if $\boldsymbol{\theta} \in \Theta_{ps}$.

The objective in Eq. 2 is similar in spirit to pseudo-likelihood objectives used for maximum likelihood estimation of parameters of Markov random fields (MRFs) [1]. The pseudo-likelihood estimate is provably consistent when the data generating distribution is a MRF of the same structure as used in the pseudo-likelihood objective. However, our setting is different since we only get to view the maximizing assignment of the MRF rather than samples from it. Thus, a particular $\boldsymbol{x}$ will always be paired with the same $\boldsymbol{y}$ rather than samples $\boldsymbol{y}$ drawn from the conditional distribution $p(\boldsymbol{y}|\boldsymbol{x}; \boldsymbol{\theta})$.

The pseudo-max constraints in Eq. 1 are also related to cutting plane approaches to inference [4, 5]. In the latter, the learning problem is solved by repeatedly looking for assignments that violate the separability constraint (or its hinge version). Our constraints can be viewed as using a very small

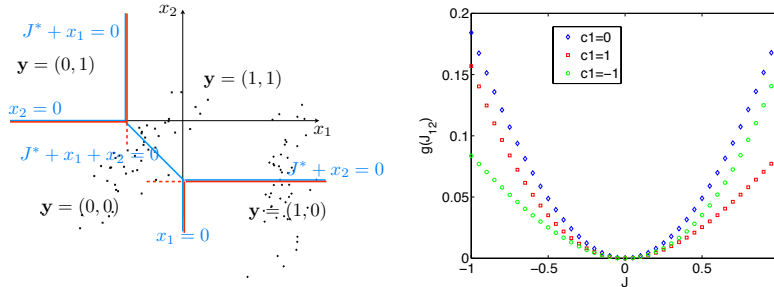

Figure 1: Illustrations for a model with two variables. **Left:** Partitioning of $\mathcal{X}$ induced by configurations $\boldsymbol{y}(\boldsymbol{x})$ for some $J^* > 0$. Blue lines carve out the exact regions. Red lines denote the pseudo-max constraints that hold with equality. Pseudo-max does not obtain the diagonal constraint coming from comparing configurations $\boldsymbol{y} = (1, 1)$ and $(0, 0)$, since these differ by more than one coordinate. **Right:** One strictly-convex component of the $\ell_{ps}(\boldsymbol{J})$ function (see Eq. 9). The function is shown for different values of $c_1$, the mean of the $x_1$ variable.

subset of assignments for the set of candidate constraint violators. We also note that when exact maximization over the discriminant function $f(\boldsymbol{y}; \boldsymbol{x}, \boldsymbol{\theta})$ is hard, the standard cutting plane algorithm cannot be employed since it is infeasible to find a violated constraint. For the pseudo-max objective, finding a constraint violation is simple and linear in the number of variables.[4]

It is easy to see (as will be elaborated on next) that the *pseudo-max* method does not in general yield a consistent estimate of $\boldsymbol{\theta}$, even in the separable case. However, as we show, consistency can be shown to be achieved under particular assumptions on the data generating distribution $p(\boldsymbol{x})$.

## 2  Consistency of the Pseudo-Max method

In this section we show that if the feature generating distribution $p(\boldsymbol{x})$ satisfies particular assumptions, then the pseudo-max approach yields a consistent estimate. In other words, if the training data is of the form $\{(\boldsymbol{x}^m, \boldsymbol{y}(\boldsymbol{x}^m; \boldsymbol{\theta}^*))\}_{m=1}^M$ for some *true* parameter vector $\boldsymbol{\theta}^*$, then as $M \to \infty$ the minimum of the pseudo-max objective will converge to $\boldsymbol{\theta}^*$ (up to equivalence transformations).

The section is organized as follows. First, we provide intuition for the consistency results by considering a model with only two variables. Then, in Sec. 2.1, we show that any parameter $\boldsymbol{\theta}^*$ can be identified to within arbitrary accuracy by choosing a particular training set (i.e., choice of $\boldsymbol{x}^m$). This in itself proves consistency, as long as there is a non-zero probability of sampling this set. In Sec. 2.2 we give a more direct proof of consistency by using strict convexity arguments.

For ease of presentation, we shall work with a simplified instance of the structured learning setting. We focus on binary variables, $y_i \in \{0, 1\}$, and consider discriminant functions corresponding to *Ising models*, a special case of pairwise MRFs ($\boldsymbol{J}$ denotes the vector of "interaction" parameters):

$$f(\boldsymbol{y}; \boldsymbol{x}, \boldsymbol{J}) = \sum_{ij \in E} J_{ij} y_i y_j + \sum_i y_i x_i \qquad (3)$$

The singleton potential for variable $y_i$ is $y_i x_i$ and is not dependent on the model parameters. We could have instead used $J_i y_i x_i$, which would be more standard. However, this would make the parameter vector $\boldsymbol{J}$ invariant to scaling, complicating the identifiability analysis. In the consistency analysis we will assume that the data is generated using a *true* parameter vector $\boldsymbol{J}^*$. We will show that as the data size goes to infinity, minimization of $\ell_{ps}(\boldsymbol{J})$ yields $\boldsymbol{J}^*$.

We begin with an illustrative analysis of the pseudo-max constraints for a model with only two variables, i.e. $f(\boldsymbol{y}; \boldsymbol{x}, J) = J y_1 y_2 + y_1 x_1 + y_2 x_2$. The purpose of the analysis is to demonstrate general principles for when pseudo-max constraints may succeed or fail. Assume that training samples are generated via $\boldsymbol{y}(\boldsymbol{x}) = \operatorname{argmax}_{\boldsymbol{y}} f(\boldsymbol{y}; \boldsymbol{x}, J^*)$. We can partition the input space $\mathcal{X}$ into four regions, $\{\boldsymbol{x} \in \mathcal{X} : \boldsymbol{y}(\boldsymbol{x}) = \hat{\boldsymbol{y}}\}$ for each of the four configurations $\hat{\boldsymbol{y}}$, shown in Fig. 1 (left). The blue lines outline the exact decision boundaries of $f(\boldsymbol{y}; \boldsymbol{x}, J^*)$, with the lines being given by the constraints

in $\Theta$ that hold with equality. The red lines denote the pseudo-max constraints in $\Theta_{ps}$ that hold with equality. For $\boldsymbol{x}$ such that $\boldsymbol{y}(\boldsymbol{x}) = (1,0)$ or $(0,1)$, the pseudo-max and exact constraints are identical.

We can identify $J^*$ by obtaining samples $\boldsymbol{x} = (x_1, x_2)$ that explore both sides of one of the decision boundaries that depends on $J^*$. The pseudo-max constraints will fail to identify $J^*$ if the samples do not sufficiently explore the transitions between $\boldsymbol{y} = (0,1)$ and $\boldsymbol{y} = (1,1)$ or between $\boldsymbol{y} = (1,0)$ and $\boldsymbol{y} = (1,1)$. This can happen, for example, when the input samples are dependent, giving only rise to the configurations $\boldsymbol{y} = (0,0)$ and $\boldsymbol{y} = (1,1)$. For points labeled $(1,1)$ around the decision line $J^* + x_1 + x_2 = 0$, pseudo-max can only tell that they respect $J^* + x_1 \geq 0$ and $J^* + x_2 \geq 0$ (dashed red lines), or $x_1 \leq 0$ and $x_2 \leq 0$ for points labeled $(0,0)$.

Only constraints that depend on the parameter are effective for learning. For pseudo-max to be able to identify $J^*$, the input samples must be continuous, densely populating the two parameter dependent decision lines that pseudo-max can use. The two point sets in the figure illustrate good and bad input distributions for pseudo-max. The diagonal set would work well with the exact constraints but badly with pseudo-max, and the difference can be arbitrarily large. However, the input distribution on the right, populating the $J^* + x_2 = 0$ decision line, would permit pseudo-max to identify $J^*$.

## 2.1 Identifiability of True Parameters

In this section, we show that it is possible to approximately identify the *true* model parameters, up to model equivalence, using the pseudo-max constraints and a carefully chosen linear number of data points. Consider the learning problem for structured prediction defined on a fixed graph $G = (V, E)$ where the parameters to be learned are pairwise potential functions $\theta_{ij}(y_i, y_j)$ for $ij \in E$ and single node fields $\theta_i(y_i)$ for $i \in V$. We consider discriminant functions of the form

$$f(\boldsymbol{y}; \boldsymbol{x}, \boldsymbol{\theta}) = \sum_{ij \in E} \theta_{ij}(y_i, y_j) + \sum_i \theta_i(y_i) + \sum_i x_i(y_i), \qquad (4)$$

where the input space $\mathcal{X} = \mathbb{R}^{|V|k}$ specifies the single node potentials. Without loss of generality, we remove the additional degrees of freedom in $\boldsymbol{\theta}$ by restricting it to be in a canonical form: $\boldsymbol{\theta} \in \Theta^{\mathrm{can}}$ if for all edges $\theta_{ij}(y_i, y_j) = 0$ whenever $y_i = 0$ or $y_j = 0$, and if for all nodes, $\theta_i(y_i) = 0$ when $y_i = 0$. As a result, assuming the training set comes from a model in this class, and the input fields $x_i(y_i)$ exercise the discriminant function appropriately, we can hope to identify $\boldsymbol{\theta}^* \in \Theta^{\mathrm{can}}$. Indeed, we show that, for some data sets, the pseudo-max constraints are sufficient to identify $\boldsymbol{\theta}^*$.

Let $\Theta_{ps}(\{\boldsymbol{y}^m, \boldsymbol{x}^m\})$ be the set of parameters that satisfy the pseudo-max classification constraints

$$\Theta_{ps}(\{\boldsymbol{y}^m, \boldsymbol{x}^m\}) = \{\boldsymbol{\theta} \mid \forall m, i, y_i \neq y_i^m, \ f(\boldsymbol{y}^m; \boldsymbol{x}^m, \boldsymbol{\theta}) \geq f(\boldsymbol{y}_{-i}^m, y_i; \boldsymbol{x}^m, \boldsymbol{\theta})\}. \qquad (5)$$

For simplicity we omit the margin losses $e(y_i^m, y_i)$, since the input fields $x_i(y_i)$ already suffice to rule out the trivial solution $\boldsymbol{\theta} = 0$.

**Proposition 2.1.** *For any $\boldsymbol{\theta}^* \in \Theta^{can}$, there is a set of $2|V|(k-1) + 2|E|(k-1)^2$ examples, $\{\boldsymbol{x}^m, \boldsymbol{y}(\boldsymbol{x}^m; \boldsymbol{\theta}^*)\}$, such that any pseudo-max consistent $\boldsymbol{\theta} \in \Theta_{ps}(\{\boldsymbol{y}^m, \boldsymbol{x}^m\}) \cap \Theta^{can}$ is arbitrarily close to $\boldsymbol{\theta}^*$.*

The proof is given in the supplementary material. To illustrate the key ideas, we consider the simpler binary discriminant function discussed in Eq. 3. Note that the binary model is already in the canonical form since $J_{ij} y_i y_j = 0$ whenever $y_i = 0$ or $y_j = 0$. For any $ij \in E$, we show how to choose two input examples $\boldsymbol{x}^1$ and $\boldsymbol{x}^2$ such that any $\boldsymbol{J}$ consistent with the pseudo-max constraints for these two examples will have $J_{ij} \in [J_{ij}^* - \epsilon, J_{ij}^* + \epsilon]$. Repeating this for all of the edge parameters then gives the complete set of examples. The input examples we need for this will depend on $\boldsymbol{J}^*$.

For the first example, we set the input fields for all neighbors of $i$ (except $j$) in such a way that we force the corresponding labels to be zero. More formally, we set $x_k^1 < -|N(k)| \max_l |J_{kl}^*|$ for $k \in N(i) \backslash j$, resulting in $y_k^1 = 0$, where $\boldsymbol{y}^1 = \boldsymbol{y}(\boldsymbol{x}^1)$. In contrast, we set $x_j^1$ to a large value, e.g. $x_j^1 > |N(j)| \max_l |J_{jl}^*|$, so that $y_j^1 = 1$. Finally, for node $i$, we set $x_i^1 = -J_{ij}^* + \epsilon$ so as to obtain a slight preference for $y_i^1 = 1$. All other input fields can be set arbitrarily. As a result, the pseudo-max constraints pertaining to node $i$ are $f(\boldsymbol{y}^1; \boldsymbol{x}^1, \boldsymbol{J}) \geq f(\boldsymbol{y}_{-i}^1, y_i; \boldsymbol{x}^1, \boldsymbol{J})$ for $y_i = 0, 1$. By taking into account the label assignments for $y_i^1$ and its neighbors, and by removing terms that are the same on both sides of the equation, we get $J_{ij} + x_i^1 + x_j^1 \geq J_{ij} y_i + y_i x_i^1 + x_j^1$, which, for $y_i = 0$, implies that $J_{ij} + x_i^1 \geq 0$ or $J_{ij} - J_{ij}^* + \epsilon \geq 0$. The second example $\boldsymbol{x}^2$ differs only in terms of the input field for $i$. In particular, we set $x_i^2 = -J_{ij}^* - \epsilon$ so that $y_i^2 = 0$. This gives $J_{ij} \leq J_{ij}^* + \epsilon$, as desired.

## 2.2 Consistency via Strict Convexity

In this section we prove the consistency of the pseudo-max approach by showing that it corresponds to minimizing a strictly convex function. Our proof only requires that $p(\boldsymbol{x})$ be non-zero for all $\boldsymbol{x} \in \mathbb{R}^n$ (a simple example being a multi-variate Gaussian) and that $\boldsymbol{J}^*$ is finite. We use a discriminant function as in Eq. 3. Now, assume the input points $\boldsymbol{x}^m$ are distributed according to $p(\boldsymbol{x})$ and that $\boldsymbol{y}^m$ are obtained via $\boldsymbol{y}^m = \arg\max_{\boldsymbol{y}} f(\boldsymbol{y}; \boldsymbol{x}^m, \boldsymbol{J}^*)$. We can write the $\ell_{ps}(\boldsymbol{J})$ objective for finite data, and its limit when $M \to \infty$, compactly as:

$$
\begin{aligned}
\ell_{ps}(\boldsymbol{J}) &= \frac{1}{M} \sum_m \sum_i \max_{y_i} \left[ (y_i - y_i^m)\big(x_i^m + \sum_{k \in N(i)} J_{ki} y_k^m\big) \right] \\
&\to \sum_i \int p(\boldsymbol{x}) \max_{y_i} \left[ (y_i - y_i(\boldsymbol{x}))\big(x_i + \sum_{k \in N(i)} J_{ki} y_k(\boldsymbol{x})\big) \right] d\boldsymbol{x}
\end{aligned} \tag{6}
$$

where $y_i(\boldsymbol{x})$ is the label of $i$ for input $\boldsymbol{x}$ when using parameters $\boldsymbol{J}^*$. Starting from the above, consider the terms separately for each $i$. We partition the integral over $\boldsymbol{x} \in \mathbb{R}^n$ into exclusive regions according to the predicted labels of the neighbors of $i$ (given $\boldsymbol{x}$). Define $S_{ij} = \{\boldsymbol{x} : y_j(\boldsymbol{x}) = 1 \text{ and } y_k(\boldsymbol{x}) = 0 \text{ for } k \in N(i)\backslash j\}$. Eq. 6 can then be written as

$$
\ell_{ps}(\boldsymbol{J}) = \sum_i \left[ \hat{g}_i(\{J_{ik}\}_{k \in N(i)}) + \sum_{k \in N(i)} g_{ik}(J_{ik}) \right], \tag{7}
$$

where $g_{ik}(J_{ik}) = \int_{\boldsymbol{x} \in S_{ik}} p(\boldsymbol{x}) \max_{y_i} [(y_i - y_i(\boldsymbol{x}))(x_i + J_{ik})] d\boldsymbol{x}$ and $\hat{g}_i(\{J_{ik}\}_{k \in N(i)})$ contains all of the remaining terms, i.e. where either zero or more than one neighbor is set to one. The function $\hat{g}_i$ is convex in $\boldsymbol{J}$ since it is a sum of integrals over convex functions. We proceed to show that $g_{ik}(J_{ik})$ is strictly convex for all choices of $i$ and $k \in N(i)$. This will show that $\ell_{ps}(\boldsymbol{J})$ is strictly convex since it is a sum over functions strictly convex in each one of the variables in $\boldsymbol{J}$.

For all values $x_i \in (-\infty, \infty)$ there is some $\boldsymbol{x}$ in $S_{ij}$. This is because for any finite $x_i$ and finite $\boldsymbol{J}^*$, the other $x_j$'s can be chosen so as to give the $\boldsymbol{y}$ configuration corresponding to $S_{ij}$. Now, since $p(\boldsymbol{x})$ has full support, we have $P(S_{ij}) > 0$ and $p(\boldsymbol{x}) > 0$ for any $\boldsymbol{x}$ in $S_{ij}$. As a result, this also holds for the marginal $p_i(x_i|S_{ij})$ over $x_i$ within $S_{ij}$. After some algebra, we obtain:

$$
g_{ij}(J_{ij}) = P(S_{ij}) \int_{-\infty}^{\infty} p_i(x_i|S_{ij}) \max[0, x_i + J_{ij}] dx_i - \int_{\boldsymbol{x} \in S_{ij}} p(\boldsymbol{x}) y_i(\boldsymbol{x})(x_i + J_{ij}) d\boldsymbol{x}
$$

The integral over the $y_i(\boldsymbol{x})(x_i + J_{ij})$ expression just adds a linear term to $g_{ij}(J_{ij})$. The relevant remaining term is (for brevity we drop $P(S_{ij})$, a strictly positive constant, and the $ij$ index):

$$
h(J) = \int_{-\infty}^{\infty} p_i(x_i|S_{ij}) \max[0, x_i + J] dx_i = \int_{-\infty}^{\infty} p_i(x_i|S_{ij}) \hat{h}(x_i, J) dx_i \tag{8}
$$

where we define $\hat{h}(x_i, J) = \max[0, x_i + J]$. Note that $h(J)$ is convex since $\hat{h}(x_i, J)$ is convex in $J$ for all $x_i$. We want to show that $h(J)$ is *strictly* convex. Consider $J' < J$ and $\alpha \in (0, 1)$ and define the interval $\mathcal{I} = [-J, -\alpha J - (1-\alpha)J']$. For $x_i \in \mathcal{I}$ it holds that: $\alpha \hat{h}(x_i, J) + (1-\alpha)\hat{h}(x_i, J') > \hat{h}(x_i, \alpha J + (1-\alpha)J')$ (since the first term is strictly positive and the rest are zero). For all other $\boldsymbol{x}$, this inequality holds but is not necessarily strict (since $\hat{h}$ is always convex in J). We thus have after integrating over $\boldsymbol{x}$ that $\alpha h(J) + (1-\alpha)h(J') > h(\alpha J + (1-\alpha)J')$, implying $h$ is strictly convex, as required. Note that we used the fact that $p(\boldsymbol{x})$ has full support when integrating over $\mathcal{I}$.

The function $\ell_{ps}(\boldsymbol{J})$ is thus a sum of strictly convex functions in all its variables (namely $g(J_{ik})$) plus other convex functions of $\boldsymbol{J}$, hence strictly convex. We can now proceed to show consistency. By strict convexity, the pseudo-max objective is minimized at a unique point $\boldsymbol{J}$. Since we know that $\ell_{ps}(\boldsymbol{J}^*) = 0$ and zero is a lower bound on the value of $\ell_{ps}(\boldsymbol{J})$, it follows that $\boldsymbol{J}^*$ is the unique minimizer. Thus we have that as $M \to \infty$, the minimizer of the pseudo-max objective is the true parameter vector, and thus we have consistency.

As an example, consider the case of two variables $y_1, y_2$, with $x_1$ and $x_2$ distributed according to $\mathcal{N}(c_1, 1), \mathcal{N}(0, 1)$ respectively. Furthermore assume $J_{12}^* = 0$. Then simple direct calculation yields:

$$
g(J_{12}) = \frac{c_1 + J_{12}}{\sqrt{2\pi}} \int_{-J_{12}-c_1}^{-c_1} e^{-x^2/2} dx - \frac{1}{\sqrt{2\pi}} e^{-c_1^2/2} + \frac{1}{\sqrt{2\pi}} e^{-(J_{12}+c_1)^2/2} \tag{9}
$$

which is indeed a strictly convex function that is minimized at $J = 0$ (see Fig. 1 for an illustration).

## 3 Hardness of Structured Learning

Most structured prediction learning algorithms use some form of inference as a subroutine. However, the corresponding prediction task is generally NP-hard. For example, maximizing the discriminant function defined in Eq. 3 is equivalent to solving Max-Cut, which is known to be NP-hard. This raises the question of whether it is possible to bypass prediction during learning. Although prediction may be intractable for arbitrary MRFs, what does this say about the difficulty of learning with a polynomial number of data points? In this section, we show that the problem of deciding whether there *exists* a parameter vector that separates the training data is NP-hard.

Put in the context of the positive results in this paper, these hardness results show that, although in some cases the pseudo-max constraints yield a consistent estimate, we cannot hope for a certificate of optimality. Put differently, although the pseudo-max constraints in the separable case always give an outer bound on $\Theta$ (and may even be a single point), $\Theta$ could be the empty set – and we would never know the difference.

**Theorem 3.1.** *Given labeled examples $\{(\boldsymbol{x}^m, \boldsymbol{y}^m)\}_{m=1}^M$ for a fixed but arbitrary graph $G$, it is NP-hard to decide whether there exists parameters $\boldsymbol{\theta}$ such that $\forall m, \boldsymbol{y}^m = \arg\max_{\boldsymbol{y}} f(\boldsymbol{y}; \boldsymbol{x}^m, \boldsymbol{\theta})$.*

*Proof.* Any parameters $\boldsymbol{\theta}$ have an equivalent parameterization in canonical form (see section Sec. 2.1, also supplementary). Thus, the examples will be separable if and only if they are separable by some $\boldsymbol{\theta} \in \Theta^{\text{can}}$. We reduce from unweighted Max-Cut. The Max-Cut problem is to decide, given an undirected graph $\mathcal{G}$, whether there exists a cut of at least $K$ edges. Let $G$ be the same graph as $\mathcal{G}$, with $k = 3$ states per variable. We construct a small set of examples where a parameter vector will exist that separates the data if and only if there is no cut of $K$ or more edges in $\mathcal{G}$.

Let $\boldsymbol{\theta}$ be parameters in canonical form equivalent to $\theta'_{ij}(y_i, y_j) = 1$ if $(y_i, y_j) \in \{(1, 2), (2, 1)\}$, 0 if $y_i = y_j$, and $-n^2$ if $(y_i, y_j) \in \{(1, 3), (2, 3), (3, 1), (3, 2)\}$. We first construct $4n + 8|E|$ examples, using the technique described in Sec. 2.1 (also supplementary material), which when restricted to the space $\Theta^{\text{can}}$, constrain the parameters to equal $\boldsymbol{\theta}$. We then use one more example $(\boldsymbol{x}^m, \boldsymbol{y}^m)$ where $\boldsymbol{y}^m = \boldsymbol{3}$ (every node is in state 3) and, for all $i$, $x_i^m(3) = \frac{K-1}{n}$ and $x_i^m(1) = x_i^m(2) = 0$. The first two states encode the original Max-Cut instance, while the third state is used to construct a labeling $\boldsymbol{y}^m$ that has value equal to $K - 1$, and is otherwise not used.

Let $K^*$ be the value of the maximum cut in $\mathcal{G}$. If in any assignment to the last example there is a variable taking the state 3 and another variable taking the state 1 or 2, then the assignment's value will be at most $K^* - n^2$, which is less than zero. By construction, the $\boldsymbol{3}$ assignment has value $K - 1$. Thus, the optimal assignment must either be $\boldsymbol{3}$ with value $K - 1$, or some combination of states 1 and 2, which has value at most $K^*$. If $K^* > K - 1$ then $\boldsymbol{3}$ is not optimal and the examples are not separable. If $K^* \leq K - 1$, the examples are separable. $\qquad\square$

This result illustrates the potential difficulty of learning in worst-case graphs. Nonetheless, many problems have a more restricted dependence on the input. For example, in computer vision, edge potentials may depend only on the difference in color between two adjacent pixels. Our results do not preclude positive results of learnability in such restricted settings. By establishing hardness of learning, we also close the open problem of relating hardness of inference and learning in structured prediction. If inference problems can be solved in polynomial time, then so can learning (using, e.g., structured perceptron). Thus, when learning is hard, inference must be hard as well.

## 4 Experiments

To evaluate our learning algorithm, we test its performance on both synthetic and real-world datasets. We show that, as the number of training samples grows, the accuracy of the pseudo-max method improves and its speed-up gain over competing algorithms increases. Our learning algorithm corresponds to solving the following, where we add $L_2$ regularization and use a scaled 0-1 loss, $e(y_i, y_i^m) = 1\{y_i \neq y_i^m\}/n_m$ ($n_m$ is the number of labels in example $m$):

$$\min_{\boldsymbol{\theta}} \frac{C}{\sum_m n_m} \sum_{m=1}^{M} \sum_{i=1}^{n_m} \max_{y_i} \left[ f(\boldsymbol{y}_{-i}^m, y_i; \boldsymbol{x}^m, \boldsymbol{\theta}) - f(\boldsymbol{y}^m; \boldsymbol{x}^m, \boldsymbol{\theta}) + e(y_i, y_i^m) \right] + \|\boldsymbol{\theta}\|^2 . \quad (10)$$

We will compare the pseudo-max method with learning using structural SVMs, both with exact inference and LP relaxations [see, e.g., 4]. We use exact inference for prediction at test time.

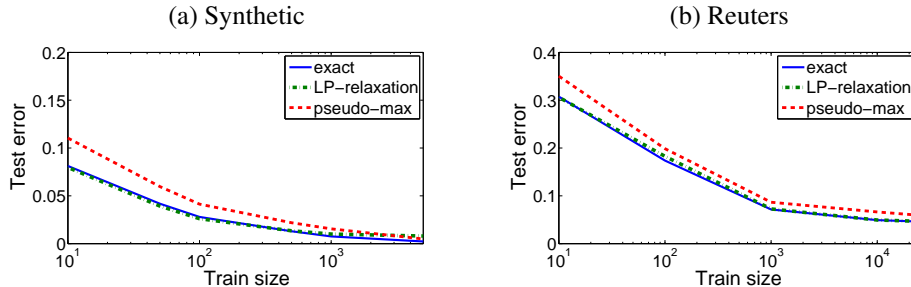

Figure 2: Test error as a function of train size for various algorithms. Subfigure (a) shows results for a synthetic setting, while (b) shows performance on the Reuters data.

In the synthetic setting we use the discriminant function $f(\boldsymbol{y}; \boldsymbol{x}, \boldsymbol{\theta}) = \sum_{ij \in E} \theta_{ij}(y_i, y_j) + \sum_i x_i \theta_i(y_i)$, which is similar to Eq. 4. We take a fully connected graph over $n = 10$ binary labels. For a weight vector $\boldsymbol{\theta}^*$ (sampled once, uniformly in the range $[-1, 1]$, and used for all train/test sets) we generate train and test instances by sampling $\boldsymbol{x}^m$ uniformly in the range $[-5, 5]$ and then computing the optimal labels $\boldsymbol{y}^m = \arg\max_{\boldsymbol{y} \in \mathcal{Y}} f(\boldsymbol{y}; \boldsymbol{x}^m, \boldsymbol{\theta}^*)$.

We generate train sets of increasing size ($M = \{10, 50, 100, 500, 1000, 5000\}$), run the learning algorithms, and measure the test error for the learned weights (with $1000$ test samples). For each train size we average the test error over $10$ repeats of sampling and training. Fig. 2(a) shows a comparison of the test error for the three learning algorithms. For small numbers of training examples, the test error of pseudo-max is larger than that of the other algorithms. However, as the train size grows, the error converges to that of exact learning, as our consistency results predict.

We also test the performance of our algorithm on a multi-label document classification task from the Reuters dataset [7]. The data consists of $M = 23149$ training samples, and we use a reduction of the dataset to the $5$ most frequent labels. The $5$ label variables form a fully connected pairwise graph structure (see [4] for a similar setting). We use random subsamples of increasing size from the train set to learn the parameters, and then measure the test error using $20000$ additional samples. For each sample size and learning algorithm, we optimize the trade-off parameter $C$ using $30\%$ of the training data as a hold-out set. Fig. 2(b) shows that for the large data regime the performance of pseudo-max learning gets close to that of the other methods. However, unlike the synthetic setting there is still a small gap, even after seeing the entire train set. This could be because the full dataset is not yet large enough to be in the consistent regime (note that exact learning has not flattened either), or because the consistency conditions are not fully satisfied: the data might be non-separable or the support of the input distribution $p(\boldsymbol{x})$ may be partial.

We next apply our method to the problem of learning the energy function for protein side-chain placement, mirroring the learning setup of [14], where the authors train a conditional random field (CRF) using tree-reweighted belief propagation to maximize a lower bound on the likelihood.[5] The prediction problem for side-chain placement corresponds to finding the most likely assignment in a pairwise MRF, and fits naturally into our learning framework. There are only $8$ parameters to be learned, corresponding to a reweighting of known energy terms. The dataset consists of $275$ proteins, where each MRF has several hundred variables (one per residue of the protein) and each variable has on average $20$ states. For prediction we use CPLEX's ILP solver.

Fig. 3 shows a comparison of the pseudo-max method and a cutting-plane algorithm which uses an LP relaxation, solved with CPLEX, for finding violated constraints.[6] We generate training sets of increasing size ($M = \{10, 50, 100, 274\}$), and measure the test error for the learned weights on the remaining examples.[7] For $M = 10, 50, 100$ we average the test error over $3$ random train/test splits, whereas for $M = 274$ we do 1-fold cross validation. We use $C = 1$ for both algorithms.

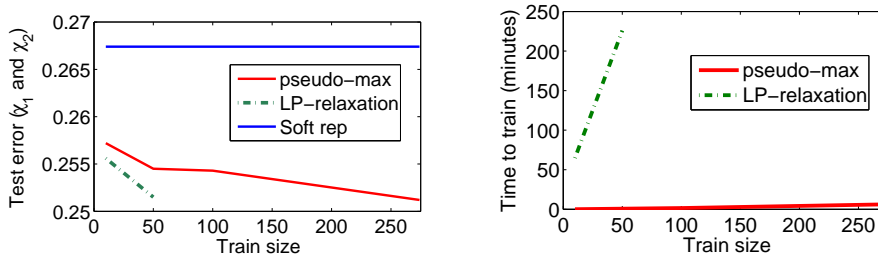

Figure 3: Training time (for one train/test split) and test error as a function of train size for both the pseudo-max method and a cutting-plane algorithm which uses a LP relaxation for inference, applied to the problem of learning the energy function for protein side-chain placement. The pseudo-max method obtains better accuracy than both the LP relaxation and HCRF (given roughly five times more data) for a fraction of the training time.

The original weights ("Soft rep" [3]) used for this energy function have 26.7% error across all 275 proteins. The best previously reported parameters, learned in [14] using a Hidden CRF, obtain 25.6% error (their training set included 55 of these 275 proteins, so this is an optimistic estimate). To get a sense of the difficulty of this learning task, we also tried a random *positive* weight vector, uniformly sampled from the range $[0, 1]$, obtaining an error of 34.9% (results would be much worse if we allowed the weights to be negative). Training using pseudo-max with 50 examples, we learn parameters in under a minute that give better accuracy than the HCRF. The speed-up of training with pseudo-max (using CPLEX's QP solver) versus cutting-plane is striking. For example, for $M = 10$, pseudo-max takes only 3 seconds, a 1000-fold speedup. Unfortunately the cutting-plane algorithm took a prohibitive amount of time to be able to run on the larger training sets. Since the data used in learning for protein side-chain placement is both highly non-separable and relatively little, these positive results illustrate the potential wide-spread applicability of the pseudo-max method.

## 5   Discussion

The key idea of our method is to find parameters that prefer the true assignment $\boldsymbol{y}^m$ over assignments that differ from it in only one variable, in contrast to *all* other assignments. Perhaps surprisingly, this weak requirement is sufficient to achieve consistency given a rich enough input distribution. One extension of our approach is to add constraints for assignments that differ from $\boldsymbol{y}^m$ in more than one variable. This would tighten the outer bound on $\Theta$ and possibly result in improved performance, but would also increase computational complexity. We could also add such competing assignments via a cutting-plane scheme so that optimization is performed only over a subset of these constraints.

Our work raises a number of important open problems: It would be interesting to derive generalization bounds to understand the convergence rate of our method, as well as understanding the effect of the distribution $p(\boldsymbol{x})$ on these rates. The distribution $p(\boldsymbol{x})$ needs to have two key properties. On the one hand, it needs to *explore* the space $\mathcal{Y}$ in the sense that a sufficient number of labels need to be obtained as the correct label for the true parameters (this is indeed used in our consistency proofs). On the other hand, $p(\boldsymbol{x})$ needs to be sufficiently sensitive close to the decision boundaries so that the true parameters can be inferred. We expect that generalization analysis will depend on these two properties of $p(\boldsymbol{x})$. Note that [11] studied active learning schemes for structured data and may be relevant in the current context.

How should one apply this learning algorithm to non-separable data sets? We suggested one approach, based on using a hinge loss for each of the pseudo constraints. One question in this context is, how resilient is this learning algorithm to label noise? Recent work has analyzed the sensitivity of pseudo-likelihood methods to model mis-specification [8], and it would be interesting to perform a similar analysis here. Also, is it possible to give any guarantees for the empirical and expected risks (with respect to exact inference) obtained by outer bound learning versus exact learning?

Finally, our algorithm demonstrates a phenomenon where more data can make computation easier. Such a scenario was recently analyzed in the context of supervised learning [12], and it would be interesting to combine the approaches.

**Acknowledgments:**   We thank Chen Yanover for his assistance with the protein data. This work was supported by BSF grant 2008303 and a Google Research Grant. D.S. was supported by a Google PhD Fellowship.

## Footnotes

[1]An alternative formulation, which we use in the next section, is to break the symmetry by having part of the input not be multiplied by any weight. This will also rule out the trivial solution $\boldsymbol{\theta} = 0$.

[2]It is possible to use $\max_i$ instead of $\sum_i$, and some of our consistency results will still hold.

[3]The pseudo-max approach is markedly different from a learning method which predicts each label $y_i$ independently, since the objective considers all $i$ simultaneously (both at learning and test time).

[4]The methods differ substantially in the non-separable setting where we minimize $\ell_{ps}(\boldsymbol{\theta})$, using a slack variable for every node and example, rather than just one slack variable per example as in $\ell(\boldsymbol{\theta})$.

[5] The authors' data and results are available from: http://cyanover.fhcrc.org/recomb-2007/

[6] We significantly optimized the cutting-plane algorithm, e.g. including a large number of initial cutting-planes and restricting the weight vector to be positive (which we know to hold at optimality).

[7] Specifically, for each protein we compute the fraction of correctly predicted $\chi_1$ and $\chi_2$ angles for all residues (except when trivial, e.g. just 1 state). Then, we compute the median of this value across all proteins.

# References

[1] J. Besag. The analysis of non-lattice data. *The Statistician*, 24:179–195, 1975.

[2] M. Collins. Discriminative training methods for hidden Markov models: Theory and experiments with perceptron algorithms. In *EMNLP*, 2002.

[3] G. Dantas, C. Corrent, S. L. Reichow, J. J. Havranek, Z. M. Eletr, N. G. Isern, B. Kuhlman, G. Varani, E. A. Merritt, and D. Baker. High-resolution structural and thermodynamic analysis of extreme stabilization of human procarboxypeptidase by computational protein design. *Journal of Molecular Biology*, 366(4):1209 – 1221, 2007.

[4] T. Finley and T. Joachims. Training structural SVMs when exact inference is intractable. In *Proceedings of the 25th International Conference on Machine Learning 25*, pages 304–311. ACM, 2008.

[5] T. Joachims, T. Finley, and C.-N. Yu. Cutting-plane training of structural SVMs. *Machine Learning*, 77(1):27–59, 2009.

[6] A. Kulesza and F. Pereira. Structured learning with approximate inference. In *Advances in Neural Information Processing Systems 20*, pages 785–792. 2008.

[7] D. Lewis, , Y. Yang, T. Rose, and F. Li. RCV1: a new benchmark collection for text categorization research. *JMLR*, 5:361–397, 2004.

[8] P. Liang and M. I. Jordan. An asymptotic analysis of generative, discriminative, and pseudolikelihood estimators. In *Proceedings of the 25th international conference on Machine learning*, pages 584–591, New York, NY, USA, 2008. ACM Press.

[9] A. F. T. Martins, N. A. Smith, and E. P. Xing. Polyhedral outer approximations with application to natural language parsing. In *ICML 26*, pages 713–720, 2009.

[10] N. Ratliff, J. A. D. Bagnell, and M. Zinkevich. (Online) subgradient methods for structured prediction. In *AISTATS*, 2007.

[11] D. Roth and K. Small. Margin-based active learning for structured output spaces. In *Proc. of the European Conference on Machine Learning (ECML)*. Springer, September 2006.

[12] S. Shalev-Shwartz and N. Srebro. SVM optimization: inverse dependence on training set size. In *Proceedings of the 25th international conference on Machine learning*, pages 928–935. ACM, 2008.

[13] B. Taskar, C. Guestrin, and D. Koller. Max margin Markov networks. In *Advances in Neural Information Processing Systems 16*, pages 25–32. 2004.

[14] C. Yanover, O. Schueler-Furman, and Y. Weiss. Minimizing and learning energy functions for side-chain prediction. *Journal of Computational Biology*, 15(7):899–911, 2008.

